# PERFORMANCE OF SYNTHETIC NEURAL NETWORK CLASSIFICATION OF NOISY RADAR SIGNALS

S. C. Ahalt        F. D. Garber        I. Jouny        A. K. Krishnamurthy

Department of Electrical Engineering
The Ohio State University, Columbus, Ohio  43210

## ABSTRACT

This study evaluates the performance of the multilayer-perceptron and the frequency-sensitive competitive learning network in identifying five commercial aircraft from radar backscatter measurements. The performance of the neural network classifiers is compared with that of the nearest-neighbor and maximum-likelihood classifiers. Our results indicate that for this problem, the neural network classifiers are relatively insensitive to changes in the network topology, and to the noise level in the training data. While, for this problem, the traditional algorithms outperform these simple neural classifiers, we feel that neural networks show the potential for improved performance.

## INTRODUCTION

The design of systems that identify objects based on measurements of their radar backscatter signals has traditionally been predicated upon decision-theoretic methods of pattern recognition [1]. While it is true that these methods are characterized by a well-defined sense of optimality, they depend on the availability of accurate models for the statistical properties of the radar measurements.

Synthetic neural networks are an attractive alternative to this problem, since they can learn to perform the classification from labeled training data, and do not require knowledge of statistical models [2]. The primary objectives of this investigation are; to establish the feasibility of using synthetic neural networks for the identification of radar objects, and to characterize the trade-offs between neural network and decision-theoretic methodologies for the design of radar object identification systems.

The present study is focused on the performance evaluation of systems operating on the received radar backscatter signals of five commercial aircraft; the Boeing 707, 727, 747, the DC-10, and the Concord. In particular, we present results for the multi–layer perceptron and the frequency-sensitive competitive learning (FSCL) synthetic network models [2,3] and compare these with results for the nearest-neighbor and maximum-likelihood classification algorithms.

In this paper, the performance of the classification algorithms is evaluated by means

of computer simulation studies; the results are compared for a number of conditions concerning the radar environment and receiver models. The sensitivity of the neural network classifiers, with respect to the number of layers and the number of hidden units, is investigated. In each case, the results obtained using the synthetic neural network classifiers are compared with those obtained using an (optimal) maximum-likelihood classifier and a (minimum-distance) nearest-neighbor classifier.

## PROBLEM DESCRIPTION

The radar system is modeled as a stepped-frequency system measuring radar backscatter at 8, 11, 17, and 28 MHz. The 8–28 MHz band of frequencies was chosen to correspond to the "resonant region" of the aircraft, i.e., frequencies with wavelengths approximately equal to the length of the object. The four specific frequencies employed for this study were pre-selected from the database maintained at The Ohio State University ElectroScience Laboratory compact radar range as the optimal features among the available measurements in this band [4].

Performance results are presented below for systems modeled as having in-phase and quadrature measurement capability (coherent systems) and for systems modeled as having only signal magnitude measurement capability (noncoherent systems). For coherent systems, the observation vector $X = [(x_1^I, x_1^Q), (x_2^I, x_2^Q), (x_3^I, x_3^Q), (x_4^I, x_4^Q)]^\mathsf{T}$ represents the in-phase and quadrature components of the noisy backscatter measurements of an unknown target. The elements of $X$ correspond to the complex scattering coefficient whose magnitude is the square root of the measured cross section (in units of square meters, $m^2$), and whose complex phase is that of the measured signal at that frequency. For noncoherent systems, the observation vector $X = [a_1, a_2, a_3, a_4]^\mathsf{T}$ consists of components which are the magnitudes of the noisy backscatter measurements corresponding to the square root of the measured cross section.

For the simulation experiments, it is assumed that the received signal is the result of a superposition of the backscatter signal vector $S$ and noise vector $W$ which is modeled as samples from an additive white Gaussian process.

## COHERENT MEASUREMENTS

In the case of a coherent radar system, the $k^{th}$ frequency component of the observation vector is given by:

$$x_k^I = (s_k^I + W_k^I), \qquad x_k^Q = (s_k^Q + W_k^Q), \tag{1}$$

where $s_k^I$ and $s_k^Q$ are the in-phase and quadrature components of the backscatter signal, and $W_k^I$ and $W_k^Q$ are the in-phase and quadrature components of the sample of the additive white Gaussian noise process at that frequency. Each of these components is modeled as a zero-mean Gaussian random variable with variance $\sigma^2/2$

so that the total additive noise contribution at each frequency is complex-valued Gaussian with zero mean and variance $\sigma^2$.

During operation, the neural network classifier is presented with the observation vector, of dimension eight, consisting of the in-phase and quadrature components of each of the four frequency measurements;

$$X = [x_1^I, x_1^Q, x_2^I, x_2^Q, x_3^I, x_3^Q, x_4^I, x_4^Q]^\mathsf{T}. \tag{2}$$

Typically, the neural net is trained using 200 samples of the observation vector $X$ for each of the five commercial aircraft discussed above. The desired output vectors are of the form

$$d_i = [d_{i,1}, \ldots, d_{i,5}] \tag{3}$$

where $d_{i,j} = 1$ for the desired aircraft and is 0 otherwise. Thus, for example, the output vector $d_i$ for the second aircraft is $0, 1, 0, 0, 0$, with a 1 appearing in the second position.

The structure of the neural nets used can be represented by $[8, n_1, \ldots, n_h, 5]$, where there are 8 input neurons, $n_i$ hidden layer neurons in the $h$ hidden layers, and 5 output neurons.

The first experiment tested the perceptron nets of varying architectures, as shown in Figures 1, and 2. As can be seen, there was little change in performance between the various nets.

The effects of the signal-to-noise ratio of the data observed during the training phase on the performance of the perceptron was also investigated. The results are presented in Figure 3. The network showed little change in performance until a training data SNR of 20 dB was reached.

We repeated this basic experiment using a winner-take-all network, the FSCL net [3]. Figure 4 shows that the performance of this network is also effected minimally by changes in network architecture.

When the FSCL net is trained with noisy data, as shown in Fig. 5, the performance decreases as the SNR of the training data increases, however, the overall performance is still very close to the performance of the multi-layer perceptron.

Our final coherent-data experiment compared the performance of the multi-layer perceptron, the FSCL net, a max-likelihood classifier and the nearest neighbor classifier. The results are shown in Figure 6. For this experiment, the training data had no superimposed noise. These results show that the max-likelihood classifier is superior, but requires full knowledge of the noise distribution. On average, the FSCL net performs better than the perceptron, but the nearest neighbor classifier performs better than either of the neural network models.

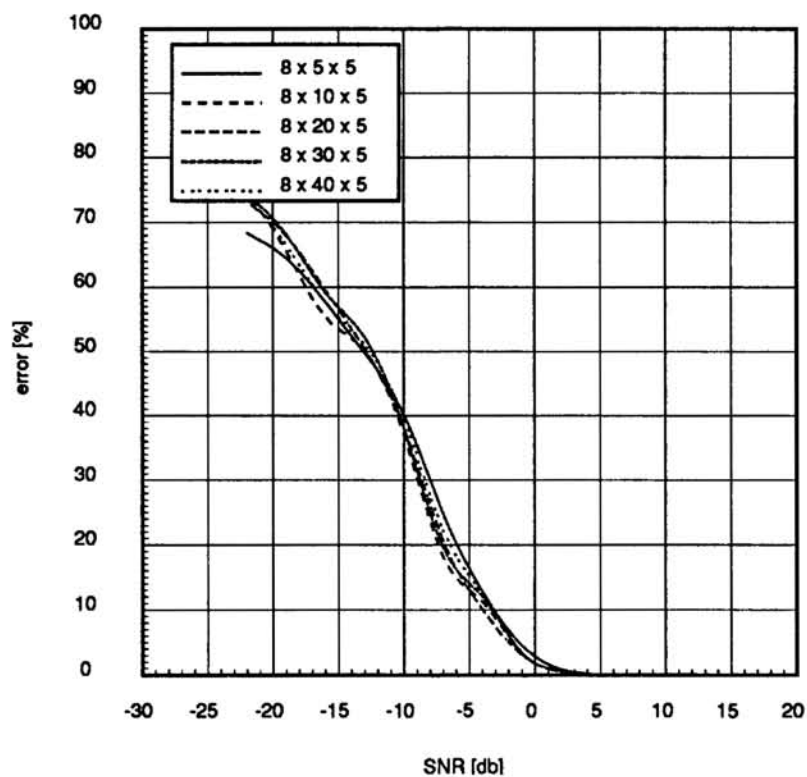

Figure 1: Performance of the perceptron with different number of hidden units.

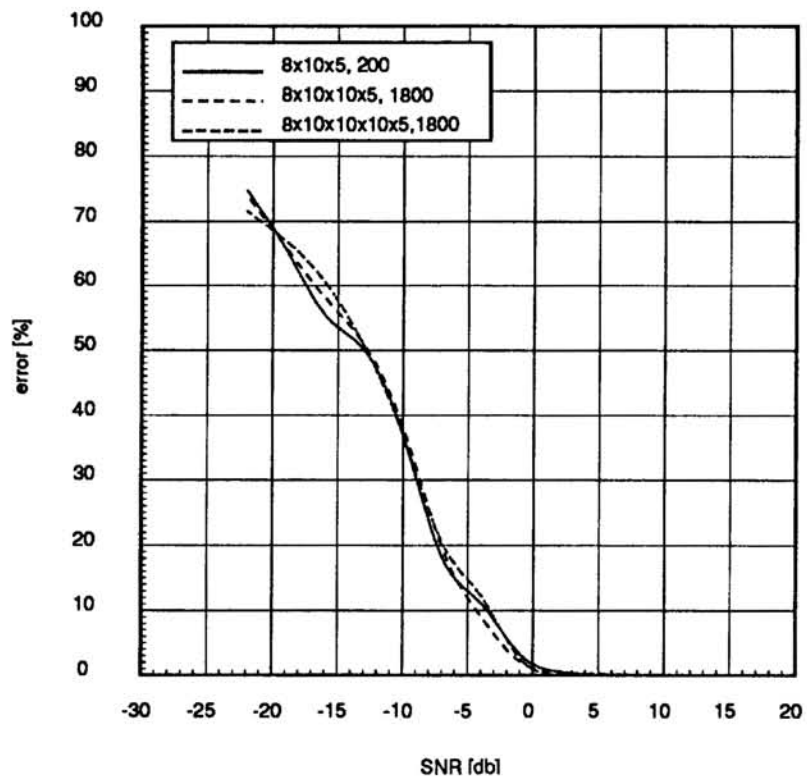

Figure 2: Performance of the perceptron with 1, 2 and 3 hidden layers.

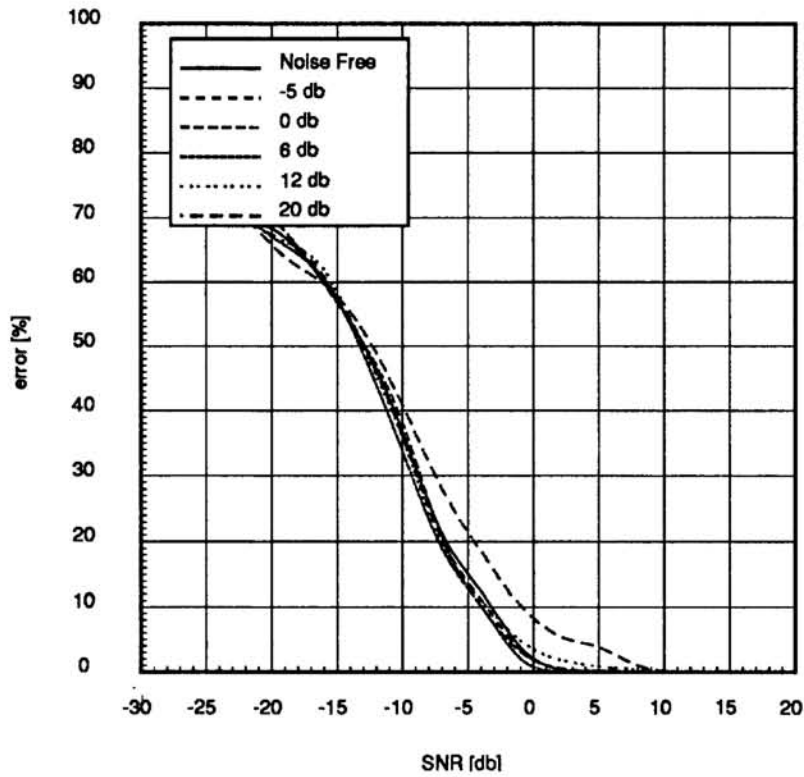

Figure 3: Performance of the perceptron for different SNR of the training data.

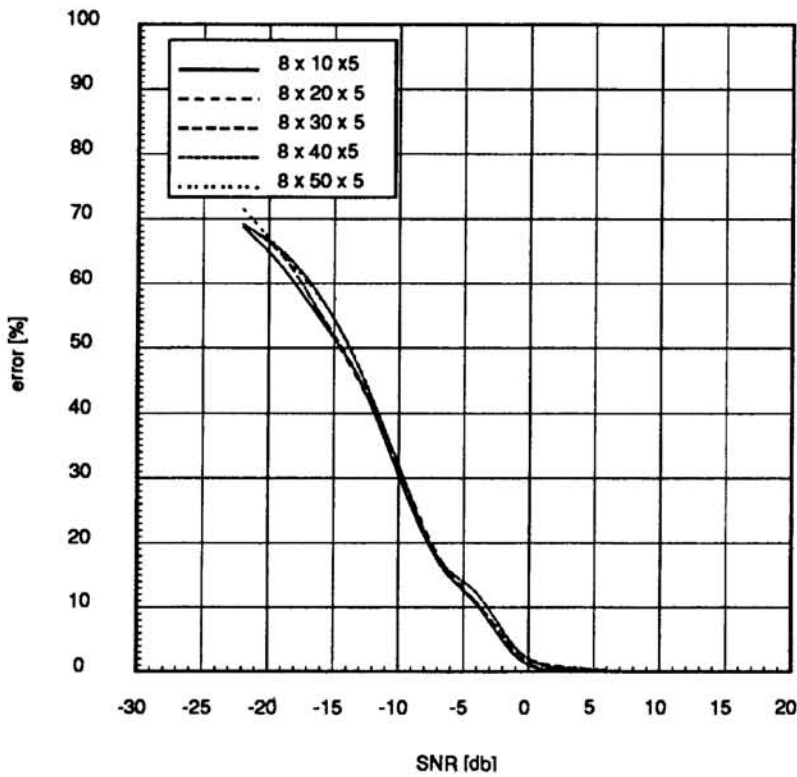

Figure 4: Performance of FSCL with varying no. of hidden units.

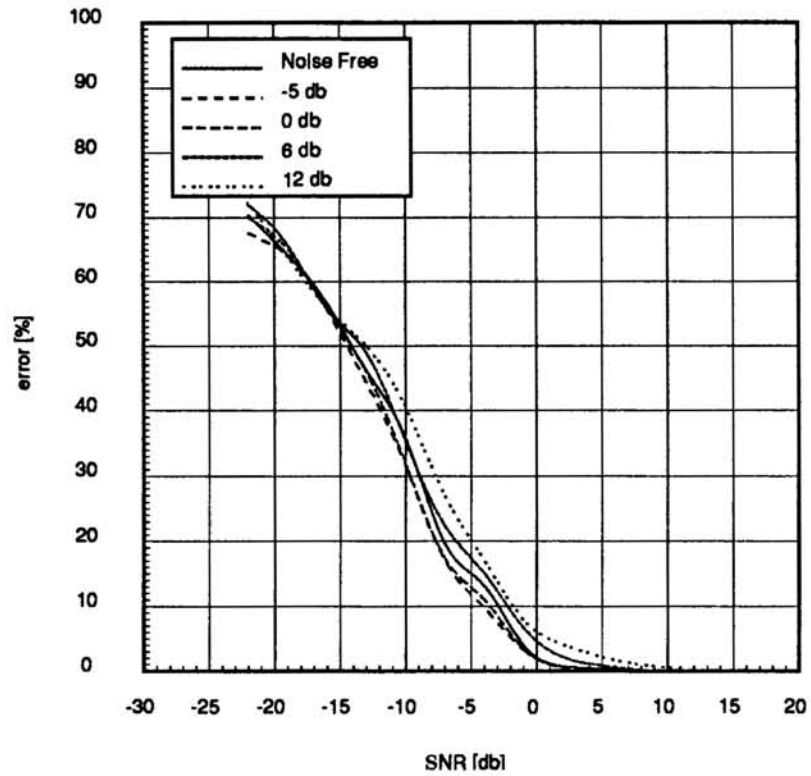

Figure 5: Performance of the FSCL network for different SNR of the training data.

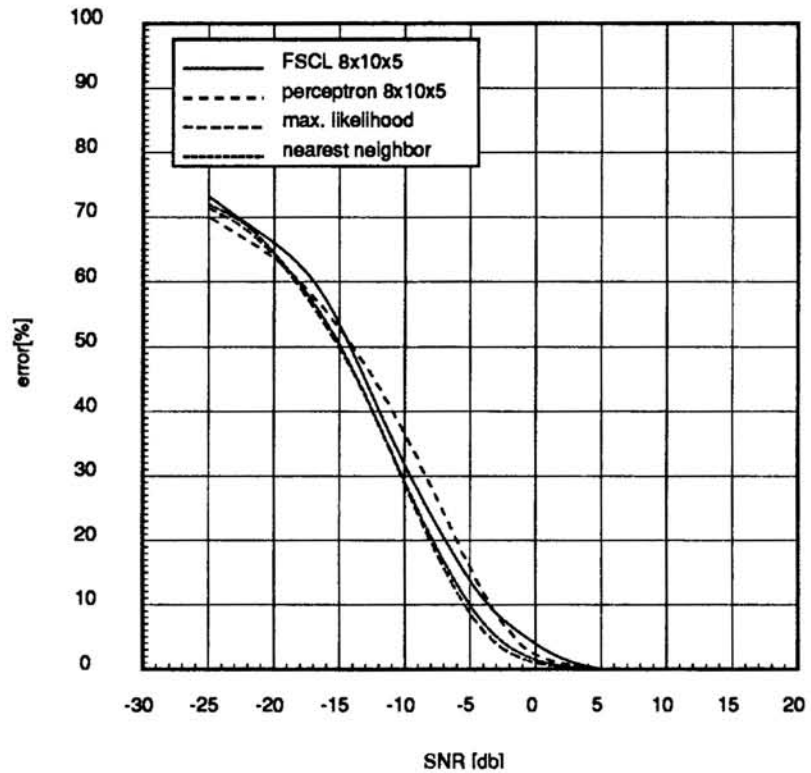

Figure 6: Comparison of all four classifiers for the coherent data case.

## NONCOHERENT MEASUREMENTS

For the case of a noncoherent radar system model, the $k^{th}$ frequency component of the observation vector is given by:

$$a_k = \sqrt{(s_k^I + W_k^I)^2 + (s_k^Q + W_k^Q)^2} \qquad (4)$$

where, as before, $s_k^I$ and $s_k^Q$ are the in-phase and quadrature components of the backscatter signal, and $W_k^I$ and $W_k^Q$ are the in-phase and quadrature components of the additive white Gaussian noise. Hence, while the underlying noise process is additive Gaussian, the resultant distribution of the observation components is Rician for the noncoherent system model.

For the case of noncoherent measurements, the neural network classifier is presented with a four-dimensional observation vector whose components are the magnitudes of the noisy measurements at each of the four frequencies;

$$X = [a_1, a_2, a_3, a_4]^\mathsf{T} \qquad (5)$$

As in the coherent case, the neural net is typically trained with 200 samples for each of the five aircraft using exemplars of the form discussed above.

The structure of the neural nets in this experiment was $[4, n_1, \ldots, n_h, 5]$ and the same training and testing procedure as in the coherent case was followed. Figure 7 shows a comparison of the performance of the neural net classifiers with both the maximum likelihood and nearest neighbor classifiers.

As before, the max-likelihood out-performs the other classifiers, with the nearest-neighbor classifier is second in performance, and the neural network classifiers perform roughly the same.

## CONCLUSIONS

These experiments lead us to conclude that neural networks are good candidates for radar classification applications. Both of the neural network learning methods we tested have a similar performance and they are both relatively insensitive to changes in network architecture, network topology, and to the noise level of the training data.

Because the methods used to implement the neural networks classifiers were relatively simple, we feel that the level of performance of the neural classifiers is quite impressive. Our ongoing research is concentrating on improving neural classifier performance by introducing more sophisticated learning algorithms such as the LVQ algorithm proposed by Kohonen [5]. We are also investigating methods of improving the performance of the perceptron, for example, by increasing training time.

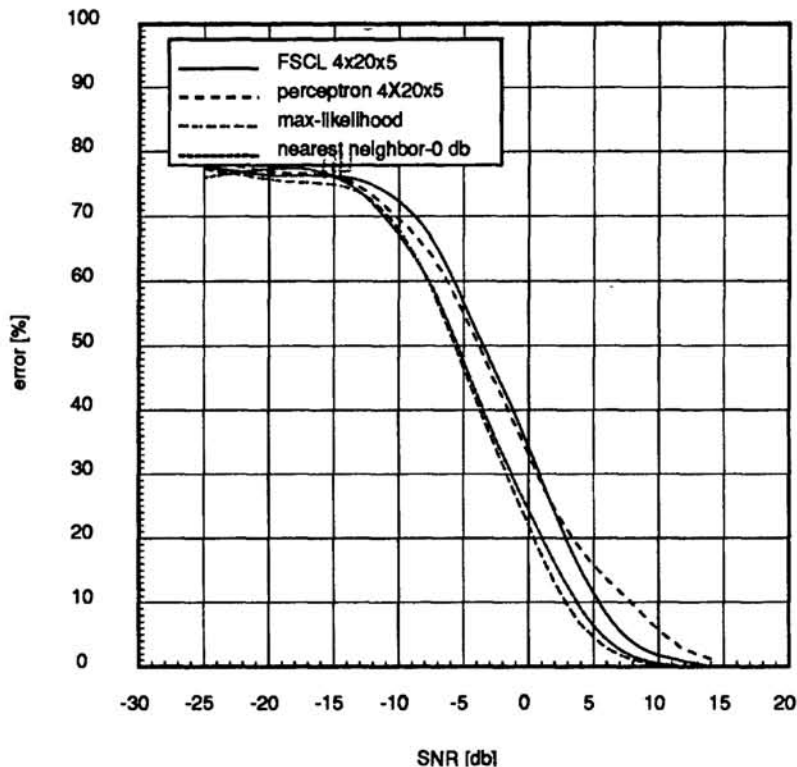

Figure 7: Comparison of all four classifiers for the noncoherent data case.

## References

[1] B. Bhanu, "Automatic target recognition: State of the art survey," *IEEE Transactions on Aerospace and Electronic Systems*, vol. AES–22, no. 4, pp. 364–379, July 1986.

[2] R. R. Lippmann, "An Introduction to Computing with Neural Nets," *IEEE ASSP Magazine*, vol. 4, no. 2, pp. 4–22, April 1987.

[3] S. C. Ahalt, A. K. Krishnamurthy, P. Chen, and D. E. Melton, "A new competitive learning algorithm for vector quantization using neural networks," *Neural Networks*, 1989. (submitted).

[4] F. D. Garber, N. F. Chamberlain, and O. Snorrason, "Time–domain and frequency–domain feature selection for reliable radar target identification," in *Proceedings of the IEEE 1988 National Radar Conference*, pp. 79–84, Ann Arbor, MI, April 20–21, 1988.

[5] T. Kohonen, *Self–Organization and Associative Memory, 2nd Ed.* Berlin: Springer–Veralg, 1988.
